# Blind channel identification for speech dereverberation using $l_1$-norm sparse learning

**Yuanqing Lin[†], Jingdong Chen[‡], Youngmoo Kim[♯], Daniel D. Lee[†]**
[†]GRASP Laboratory, Department of Electrical and Systems Engineering, University of Pennsylvania
[‡]Bell Laboratories, Alcatel-Lucent
[♯] Department of Electrical and Computer Engineering, Drexel University

## Abstract

Speech dereverberation remains an open problem after more than three decades of research. The most challenging step in speech dereverberation is *blind channel identification* (BCI). Although many BCI approaches have been developed, their performance is still far from satisfactory for practical applications. The main difficulty in BCI lies in finding an appropriate acoustic model, which not only can effectively resolve solution degeneracies due to the lack of knowledge of the source, but also robustly models real acoustic environments. This paper proposes a *sparse acoustic room impulse response (RIR) model* for BCI, that is, an acoustic RIR can be modeled by a *sparse* FIR filter. Under this model, we show how to formulate the BCI of a single-input multiple-output (SIMO) system into a $l_1$-norm regularized least squares (LS) problem, which is *convex* and can be solved efficiently with guaranteed global convergence. The sparseness of solutions is controlled by $l_1$-norm regularization parameters. We propose a *sparse learning* scheme that infers the optimal $l_1$-norm regularization parameters directly from microphone observations under a Bayesian framework. Our results show that the proposed approach is effective and robust, and it yields source estimates in real acoustic environments with high fidelity to anechoic chamber measurements.

## 1   Introduction

Speech dereverberation, which may be viewed as a denoising technique, is crucial for many speech related applications, such as hands-free teleconferencing and automatic speech recognition. It is a challenging signal processing task and remains an open problem after more than three decades of research. Although many approaches [1] have been developed for speech dereverberation, blind channel identification (BCI) is believed to be the key to thoroughly solving the dereverberation problem. Most BCI approaches rely on source statistics (higher order statistics [2] or statistics of LPC coefficients [3]), or spatial difference among multiple channels [4] for resolving solution degeneracies due to the lack of knowledge of the source. The performance of these approaches depends on how well they model real acoustic systems (mainly sources and channels). The BCI approaches using source statistics need a long sequence of data to build up the statistics, and their performance often degrades significantly in real acoustic environments where acoustic systems are time-varying and only approximately time-invariant during a short time window. Besides the data efficiency issue, there are some other difficulties in the BCI approaches using source statistics, for example, non-stationarity of a speech source, whitening side effect, and non-minimum phase of a filter [2]. In contrast, the BCI approaches exploiting channel spatial difference are blind to the source, and thus they avoid those difficulties arising in assuming source statistics. Unfortunately, these approaches are often too ill-conditioned to tolerate even a very small amount of ambient noise. In general, BCI for speech dereverberation is an active research area, and the main challenge is how to build an effective acoustic model that not only can resolve solution degeneracies due to the lack of knowledge of the source, but also robustly models real acoustic environments.

To address the challenge, this paper proposes a *sparse acoustic room impulse response (RIR) model* for BCI, that is, an acoustic RIR can be modeled by a *sparse* FIR filter. The sparse RIR model is theoretically sound [5], and it has been shown to be useful for estimating RIRs in real acoustic environments when the source is given *a priori* [6]. In this paper, the sparse RIR model is incorporated with channel spatial difference, resulting a *blind sparse channel identification* (BSCI) approach for a single-input multiple-output (SIMO) acoustic system. The BSCI approach aims to resolve some of the difficulties in conventional BCI approaches. It is blind to the source and therefore avoids the difficulties arising in assuming source statistics. Meanwhile, the BSCI approach is expected to be robust to ambient noise. It has been shown that, when the source is given *a priori* [7], the prior knowledge about sparse RIRs plays an important role in robustly estimating RIRs in noisy acoustic environments. Furthermore, the statistics describing the sparseness of RIRs are governed by acoustic room characteristics, and thus they are close to be stationary with respect to a specific room. This is advantageous in terms of both learning the statistics and applying them in channel identification.

Based on the cross relation formulation [4] of BCI, this paper develops a BSCI algorithm that incorporates the sparse RIR model. Our choice for enforcing sparsity is $l_1$-norm regularization [8], which has been the driving force for many emerging fields in signal processing, such as sparse coding and compressive sensing. In the context of BCI, two important issues need to be addressed when using $l_1$-norm regularization. First, the existing cross relation formulation for BCI is nonconvex, and directly enforcing $l_1$-norm regularization will result in an intractable optimization. Second, $l_1$-norm regularization parameters are critical for deriving correct solutions, and their improper setting may lead to totally irrelevant solutions. To address these two issues, this paper shows how to formulate the BCI of a SIMO system into a *convex* optimization, indeed an unconstrained least squares (LS) problem, which provides a flexible platform for incorporating $l_1$-norm regularization; it also shows how to infer the *optimal* $l_1$-norm regularization parameters directly from microphone observations under a Bayesian framework.

We evaluate the proposed BSCI approach using both simulations and experiments in real acoustic environments. Simulation results illustrate the effectiveness of the proposed sparse RIR model in resolving solution degeneracies, and they show that the BSCI approach is able to robustly and accurately identify filters from noisy microphone observations. When applied to speech dereverberation in real acoustic environments, the BSCI approach yields source estimates with high fidelity to anechoic chamber measurements. All of these demonstrate that the BSCI approach has the potential for solving the difficult speech dereverberation problem.

## 2 Blind sparse channel identification (BSCI)

### 2.1 Previous work

Our BSCI approach is based on the cross relation formulation for blind SIMO channel identification [4]. In a one-speaker two-microphone system, the microphone signals at time $k$ can be written as:

$$x_i(k) = s(k) * h_i + n_i(k), \ \ i = 1, 2, \tag{1}$$

where $*$ denotes linear convolution, $s(k)$ is a source signal, $h_i$ represents the channel impulse response between the source and the $i$th microphone, and $n_i(k)$ is ambient noise. The cross relation formulation is based on a clever observation, $x_2(k) * h_1 = x_1(k) * h_2 = s(k) * h_1 * h_2$, if the microphone signals are noiseless [4]. Then, without requiring any knowledge from the source signal, the channel filters can be identified by minimizing the squared cross relation error. In matrix-vector form, the optimization can be written as

$$\mathbf{h}_1^*, \mathbf{h}_2^* = \underset{\|\mathbf{h}_1\|^2 + \|\mathbf{h}_2\|^2 = 1}{\operatorname{argmin}} \frac{1}{2} \|\mathbf{X}_2 \mathbf{h}_1 - \mathbf{X}_1 \mathbf{h}_2\|^2 \tag{2}$$

where $\mathbf{X}_i$ is the $(N + L - 1) \times L$ convolution Toeplitz matrix whose first row and first column are $[x_i(k - N + 1), 0, \ldots, 0]$ and $[x_i(k - N + 1), x_i(k - N + 2), ..., x_i(k), 0, \ldots, 0]^T$, respectively, $N$ is the microphone signal length, $L$ is the filter length, $\mathbf{h}_i(i = 1, 2)$ are $L \times 1$ vectors representing the filters, $\| \cdot \|$ denotes $l_2$-norm, and the constraint is to avoid the trivial zero solution. It is easy to see that the above optimization is a minimum eigenvalue problem, and it can be solved by eigenvalue decomposition. As shown in [4], the eigenvalue decomposition approach finds the true solution within a constant time delay and a constant scalar factor when 1) the system is noiseless; 2) the two

filters are co-prime (namely, no common zeros); and 3) the system is sufficiently excited (i.e., the source needs to have enough frequency bands).

Unfortunately, the eigenvalue decomposition approach has not been demonstrated to be useful for speech dereverberation in real acoustic environments. This is because the conditions for finding true solutions are difficult to sustain. First, microphone signals in real acoustic environments are always immersed in excessive ambient noise (such as air-conditioning noise), and thus the noiseless assumption is never true. Second, it requires precise information about filter order for the filters to be co-prime, however, the filter order itself is hard to compute accurately since the filters modeling RIRs are often thousands of taps long. As a result, eigenvalue decomposition approach is often ill-conditioned and very sensitive to even a very small amount of ambient noise.

Our proposed sparse RIR model aims to alleviate those difficulties. Under the sparse RIR model, sparsity regularization automatically determines filter order since surplus filter coefficients are forced to be zero. Furthermore, previous work [7] has demonstrated that, when the source is given *a priori*, sparsity regularization plays an important role in robustly estimating RIRs in noisy acoustic environments. In order to exploit the sparse RIR model, we first formulate the BCI using cross relation into a *convex* optimization, which will provide a flexible platform for enforcing $l_1$-norm sparsity regularization.

## 2.2 Convex formulation

The optimization in Eq. 2 is nonconvex because its domain, $\|\mathbf{h}_1\|^2 + \|\mathbf{h}_2\|^2 = 1$, is nonconvex. We propose to replace it with a *convex* singleton linear constraint, and the optimization becomes

$$\mathbf{h}_1^*, \mathbf{h}_2^* = \operatorname*{argmin}_{h_1(l)=1} \frac{1}{2}\|\mathbf{X}_2\mathbf{h}_1 - \mathbf{X}_1\mathbf{h}_2\|^2 \tag{3}$$

where $h_1(l)$ is the $l$th element of filter $\mathbf{h}_1$. It is easy to see that, when microphone signals are noiseless, the optimizations in Eqs. 2 and 3 yield equivalent solutions within a constant time delay and a constant scalar factor. Because the optimization is a minimization, $h_1(l)$ tends to align with the largest coefficient in filter $\mathbf{h}_1$, which normally is the coefficient corresponding to the direct path. Consequently, the singleton linear constraint removes two degrees of freedom in filter estimates: a constant time delay (by fixing $l$) and a constant scalar factor [by fixing $h_1(l) = 1$]. The choice of $l$ ($0 \leq l \leq L - 1$) is arbitrary as long as the direct path in filter $\mathbf{h}_2$ is no more than $l$ samples earlier than the one in filter $\mathbf{h}_1$.

The new formulation in Eq. 3 has many advantages. It is convex and indeed an unconstrained LS problem since the singleton linear constraint can be easily substituted into the objective function. Furthermore, the new LS formulation is more robust to ambient noise than the eigenvalue decomposition approach in Eq. 2. This can be better viewed in the frequency domain. Because the squared cross relation error (the objective function in Eqs. 2 and 3) is weighted in the frequency domain by the power spectrum density of a common source, the total filter energy constraint in Eq. 2 may be filled with less significant frequency bands which contribute little to the source and are weighted less in the objective function. As a result, the eigenvalue decomposition approach is very sensitive to noise. In contrast, the singleton linear constraint in Eq. 3 has much less coupling in filter energy allocation, and the new LS approach is more robust to ambient noise.

Then, the BSCI approach is to incorporate the LS formulation with $l_1$-norm sparsity regularization, and the optimization becomes

$$\mathbf{h}_1^*, \mathbf{h}_2^* = \operatorname*{argmin}_{h_1(l)=1} \frac{1}{2}\|\mathbf{X}_2\mathbf{h}_1 - \mathbf{X}_1\mathbf{h}_2\|^2 + \lambda' \sum_{j=0}^{L-1}[|h_1(j)| + |h_2(j)|] \tag{4}$$

where $\lambda'$ is a nonnegative scalar regularization parameter that balances the preference between the squared cross relation error and the sparseness of solutions described by their $l_1$-norm. The setting of $\lambda'$ is critical for deriving appropriate solutions, and we will show how to compute its optimal setting in a Bayesian framework in Section 2.3. Given a $\lambda'$, the optimization in Eq. 4 is *convex* and can be solved by various methods with guaranteed global convergence. We implemented the *Mehrotra predictor-corrector primal-dual interior point method* [9], which is known to yield better search directions than the Newton's method. Our implementation usually solves the optimization in Eq. 4 with extreme accuracy (relative duality gap less than $10^{-14}$) in less than 20 iterations.

## 2.3 Bayesian $l_1$-norm sparse learning for blind channel identification

The $l_1$-norm regularization parameter $\lambda'$ in Eq. 4 is critical for deriving appropriately sparse solutions. How to determine its optimal setting is still an open research topic. A recent development is to solve the optimization in Eq. 4 with respect to all possible values of $\lambda'$ [10], and cross-validation is then employed to find an appropriate solution. However, it is not easy to obtain extra data for cross-validation in BCI since real acoustic environments are often time-varying. In this study, we develop a Bayesian framework for inferring the *optimal* regularization parameters for the BSCI formulation in Eq. 4. A similar Bayesian framework can be found in [7], where the source was assumed to be known *a priori*.

The optimization in Eq. 4 is a *maximum-a-posteriori* estimation under the following probabilistic assumptions

$$P\left(\mathbf{X}_2\mathbf{h}_1 - \mathbf{X}_1\mathbf{h}_2|\sigma^2, \mathbf{h}_1, \mathbf{h}_2\right) = \frac{1}{(2\pi\sigma^2)^{(N+L-1)/2}}\exp\left\{-\frac{1}{2\sigma^2}\|\mathbf{X}_2\mathbf{h}_1 - \mathbf{X}_1\mathbf{h}_2\|^2\right\}, \quad (5)$$

$$P\left(\mathbf{h}_1, \mathbf{h}_2|\lambda\right) = \left(\frac{\lambda}{2}\right)^{2L}\exp\left\{-\lambda\sum_{j=0}^{L-1}[|h_1(j)| + |h_2(j)|]\right\} \quad (6)$$

where the cross relation error is an I.I.D. zero-mean Gaussian with variance $\sigma^2$, and the filter coefficients are governed by a Laplacian sparse prior with the scalar parameter $\lambda$. Then, the regularization parameter $\lambda'$ in Eq. 4 can be written as

$$\lambda' = \sigma^2\lambda. \quad (7)$$

When the ambient noise [$n_1(k)$ and $n_2(k)$ in Eq. 1] is an I.I.D. zero-mean Gaussian with variance $\sigma_0^2$, the parameter $\sigma^2$ can be approximately written as

$$\sigma^2 = \sigma_0^2(\|\mathbf{h}_1\|^2 + \|\mathbf{h}_2\|^2), \quad (8)$$

because $x_2(k) * h_1 - x_1(k) * h_2 = n_2(k) * h_1 - n_1(k) * h_2$. The above form of $\sigma^2$ is only an approximation because the cross relation error is temporally correlated through the convolution. Nevertheless, since the cross relation error is the result of the convolutive mixing, its distribution will be close to the Gaussian with its variance described by Eq. 8 according to the central limit theorem. We choose to estimate the ambient noise level ($\sigma_0^2$) directly from microphone observations via restricted maximum likelihood [11]:

$$\sigma_0^2 = \min_{\mathbf{s}, \mathbf{h}_1, \mathbf{h}_2} \frac{1}{N-L-1} \sum_{i=1}^{2}\sum_{k=0}^{N-1}\|x_i(k) - s(k) * h_i\|^2 \quad (9)$$

where the denominator $N - L - 1$ (but not $2N$) accounts for the loss of the degrees of freedom during the optimization. The above minimization is solved by coordinate descent alternatively with respect to the source and the filters. It is initialized with the LS solution by Eq. 3 and often able to yield a good $\sigma_0^2$ estimate in a few iterations. Note that each iteration can be computed efficiently in the frequency domain. Meanwhile, the parameter $\lambda$ can be computed by

$$\lambda = \frac{2L}{\sum_{j=0}^{L-1}[|h_1(j)| + |h_2(j)|]}, \quad (10)$$

as a result of finding the optimal Laplacian distribution given its sufficient statistics.

With the Eqs. 8 and 10, finding the optimal regularization parameters becomes computing the statistics of filters, $\|\mathbf{h}_1\|^2 + \|\mathbf{h}_2\|^2$ and $\sum_{j=0}^{L-1}[|h_1(j)| + |h_2(j)|]$. These statistics are closely related to acoustic room characteristics and may be computed from them if they are known *a priori*. For example, the reverberation time of a room defines how fast echoes decay $-60$ dB, and it can be used to compute the filter statistics. More generally, we choose to compute the statistics directly from microphone observations in the Baysian framework by maximizing the *marginal likelihood*, $P(\mathbf{X}_2\mathbf{h}_1 - \mathbf{X}_1\mathbf{h}_2|\sigma^2, \lambda) = \int_{h_1(l)=1} P(\mathbf{X}_2\mathbf{h}_1 - \mathbf{X}_1\mathbf{h}_2, \mathbf{h}_1, \mathbf{h}_2|\sigma^2, \lambda)\mathbf{dh}_1\mathbf{dh}_2$. The optimization is through Expectation-Maximization (EM) updates [7]:

$$\sigma^2 \longleftarrow \sigma_0^2\int_{h(l)=1}(\|\mathbf{h}_1\|^2 + \|\mathbf{h}_2\|^2)Q(\mathbf{h}_1, \mathbf{h}_2)\mathbf{dh}_1\mathbf{dh}_2 \quad (11)$$

$$\lambda \longleftarrow \frac{2L}{\int_{h(l)=1}(\sum_{j=0}^{L-1}|h_1(j)| + |h_2(j)|)Q(\mathbf{h}_1, \mathbf{h}_2)\mathbf{dh}_1\mathbf{dh}_2} \quad (12)$$

where $\mathbf{h}_1$ and $\mathbf{h}_2$ are treated as hidden variables, $\sigma^2$ and $\lambda$ are parameters, and $Q(\mathbf{h}_1, \mathbf{h}_2) \propto \exp\{-\frac{1}{2\sigma^2}\|\mathbf{X}_2\mathbf{h}_1 - \mathbf{X}_1\mathbf{h}_2\|^2 - \lambda[\sum_{j=0}^{L-1} |h_1(j)| + |h_2(j)|]\}$ is the probability distribution of $\mathbf{h}_1$ and $\mathbf{h}_2$ given the current estimate of $\sigma^2$ and $\lambda$. The integrals in Eqs. 11 and 12 can be computed using the variational scheme described in [7]. The EM updates often converge to a good estimate of $\sigma^2$ and $\lambda$ in a few iterations. Moreover, since the filter statistics are relatively stationary for a specified room, the Bayesian inference may be carried out off-line and only once if the room conditions stay the same.

After the filters are identified by BCI approaches, the source can be computed by various methods [12]. We choose to estimate the source by the following optimization

$$\mathbf{s}^* = \underset{\mathbf{s}}{\operatorname{argmin}} \sum_{i=1}^{2} \sum_{k=0}^{N-1} \|x_i(k) - s(k) * h_i\|^2, \tag{13}$$

which will yield maximum-likelihood (ML) estimation if the filter estimates are accurate.

## 3 Simulations and Experiments

### 3.1 Simulations

#### 3.1.1 Simulations with artificial RIRs

We first employ a simulated example to illustrate the effectiveness of the proposed sparse RIR model for BCI. In the simulation, we used a speech sequence of 1024 samples (with 16 kHz sampling rate) as the source ($s$) and simulated two 16-sample FIR filters ($h_1$ and $h_2$). The filter $h_1$ had nonzero elements only at indices 0, 2, and 12 with amplitudes of 1, -0.7, and 0.5, respectively; the filter $h_2$ had nonzero elements only at indices 2, 6, 8, and 10 with amplitudes of 1, -0.6, 0.6, and 0.4, respectively. Notice that both $h_1$ and $h_2$ are sparse. Then the simulated microphone observations ($x_1$ and $x_2$) were computed by Eq. 1 with the ambient noise being real noise recorded in a classroom. The noise was scaled so that the signal-to-noise ratio (SNR) of the microphone signals was approximately 20 dB. Because a big portion of the noise (mainly air-conditioning noise) was at low frequency, the microphone observations were high-passed with a cut-off frequency of 100 Hz before they were fed to BCI algorithms. In the BSCI algorithm, the $l_1$-norm regularization parameters, $\sigma^2$ and $\lambda$, were estimated in the Bayesian framework using the update rules given in Eqs. 11 and 12.

Figure 1 shows the filters identified by different BCI approaches. Compared to the conventional eigenvalue decomposition method (Eq. 2), the new convex LS approach (Eq. 3) is more robust to ambient noise and yielded better filter estimates even though the estimates still seem to be convolved by a common filter. The proposed BSCI approach (Eq. 4) yielded filter estimates that are almost identical to the true ones. It is evident that the proposed sparse RIR model played a crucial role in robustly and accurately identifying filters in blind manners. The robustness and accuracy gained by the BSCI approach will become essential when the filters are thousands of taps long in real acoustic environments.

#### 3.1.2 Simulations with measured RIRs

Here we employ simulations using RIRs measured in real rooms to demonstrate the effectiveness of the proposed BSCI approach for speech dereverberation. Its performance is compared to the beamforming, the eigenvalue decomposition (Eq. 2), and the LS (Eq. 3) approaches. In the simulation, the source sequence ($s$) was a sentence of speech (approximately 1.5 seconds), and the filters ($h_1$ and $h_2$) were two measured RIRs from York MARDY database (http://www.commsp.ee.ic.ac.uk/ sap/mardy.htm) but down-sampled to 16 kHz (from originally 48 kHz). The original filters in the database were not sparse, but they had many tiny coefficients which were in the range of measurement uncertainty. To make the simulated filters sparse, we simply zeroed out those coefficients whose amplitudes were less than 2% of the maximum. Finally, we truncated the filters to have length of 2048 since there were very few nonzero coefficients after that. With the simulated source and filters, we then computed microphone observations using Eq. 1 with ambient noise being real noise recorded in a classroom. For testing the robustness of different BCI algorithms, the ambient noise was scaled to different levels so that the SNRs varied from 60 dB to 10 dB. Similar to the previous simulations, the simulated observations were high-passed with a cutoff

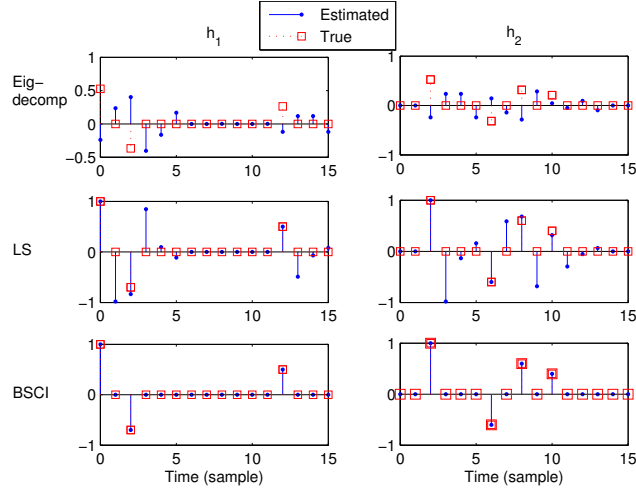

Figure 1: Identified filters by three different BCI approaches in a simulated example: the eigenvalue decomposition approach (denoted as eig-decomp) in Eq. 2, the LS approach in Eq. 3, and the blind sparse channel identification (BSCI) approach in Eq. 4. The solid-dot lines represent the estimated filters, and the dot-square lines indicate the true filters within a constant time delay and a constant scalar factor.

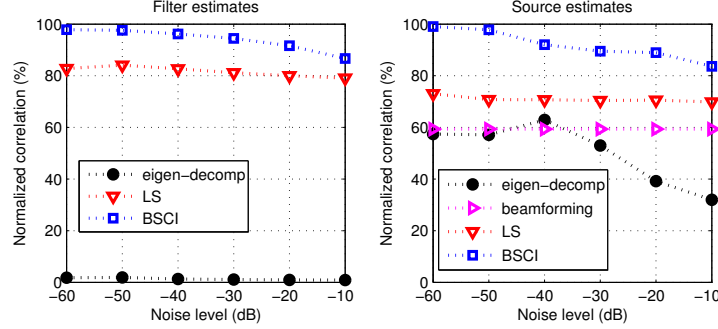

Figure 2: The simulation results using measured real RIRs. The normalized correlation (defined in Eq. 14) of the estimates were computed with respect to their true values. The filters were identified by three different approaches: the eigenvalue decomposition approach (denoted as eigen-decomp) in Eq. 2 , the LS approach in Eq. 3, and the blind sparse channel identification (BSCI) approach in Eq. 4. After the filters were identified, the source was estimated by Eq. 13. The source estimated by beamforming is also presented as a baseline reference.

frequency of 100 Hz before they were fed to different BCI algorithms. In the BSCI approach, the $l_1$-norm regularization parameters were iteratively computed using the updates in Eqs. 11 and 12. After filters were identified, the source was estimated using Eq. 13.

Because both filter and source estimates by BCI algorithms are within a constant time delay and a constant scalar factor, we use normalized correlation for evaluating the estimates. Let $\hat{\mathbf{s}}$ and $\mathbf{s_0}$ denote an estimated source and the true source, respectively, then the normalized correlation $C(\hat{\mathbf{s}}, \mathbf{s}_0)$ is defined as

$$C(\hat{\mathbf{s}}, \mathbf{s}_0) = \max_m \frac{\sum_k \hat{s}(k - m)s_0(k)}{\|\hat{\mathbf{s}}\|\|\mathbf{s}_0\|} \quad (14)$$

where $m$ and $k$ are sample indices, and $\|\cdot\|$ denotes $l_2$-norm. It is easy to see that, the normalized correlation is between $0\%$ and $100\%$: it is equal to $0\%$ when the two signals are uncorrelated, and it is equal to $100\%$ only when the two signal are identical within a constant time delay and a constant scalar factor. The definition in Eq. 14 is also applicable to the evaluation of filter estimates.

The simulation results are shown in Fig. 2. Similar to what we observed in the previous example, the convex LS approach (Eq. 3) shows significant improvement in both filter and source estimation compared to the eigenvalue decomposition approach (Eq. 2). In fact, the eigenvalue decomposition

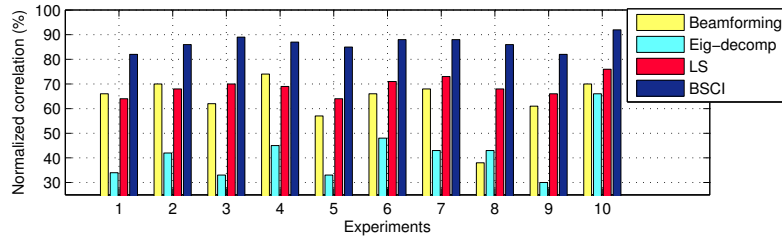

Figure 3: The source estimates of 10 experiments in real acoustic environments. The normalized correlation was with respect to their anechoic chamber measurement. The filters were identified by three different BCI approaches: the eigenvalue decomposition approach (denoted as eig-decomp) in Eq. 2, the LS approach in Eq. 3, and the blind sparse channel identification (BSCI) approach in Eq. 4. The beamforming results serve as the baseline performance for comparison.

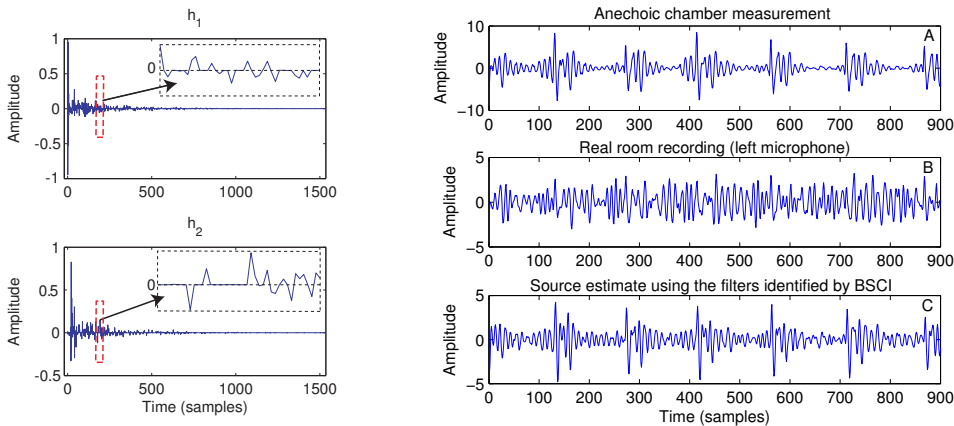

Figure 4: Results of Experiment 6 in Fig. 3. Left: the filters estimated by the proposed blind sparse channel identification (BSCI) approach. They are sparse as indicated by the enlarged segments. Right: a segment of source estimate (shown in C) using the BSCI approach. It is compared with its anechoic measurement (shown in A) and its microphone recording (shown in B).

approach did not yield relevant results because it was too ill-conditioned due to the long filters. The remarkable performance came from the BSCI approach, which incorporates the convex LS formulation with the sparse RIR model. In particular, the BSCI approach yielded higher than 90% normalized correlation in source estimates when SNR was better than 20 dB, and it yielded higher than 99% normalized correlation in the low noise limit. The performance of the canonical delay-and-sum beamforming is also presented as the baseline for all BCI algorithms.

### 3.2 Experiments

We also evaluated the proposed BSCI approach using signals recorded in real acoustic environments. We carried out 10 experiments in total in a reverberant room. In each experiment, a sentence of speech (approximately 1.5 seconds, and the same for all experiments) was played through a loudspeaker (NSW2-326-8A, Aura Sound) and recorded by a matched omnidirectional microphone pair (M30MP, Earthworks). The speaker-microphone positions (and thus RIRs) were different in different experiments. Because the recordings had a large amount of low-frequency noise, they were high-passed with a cutoff frequency of 100 Hz before they were fed to BCI algorithms. In the BSCI approach, the $l_1$-norm regularization parameters, $\sigma^2$ and $\lambda$, were iteratively computed using the updates in Eq. 11 and 12. After the filters were identified, the sources were computed using Eq. 13. We also had recordings in the anechoic chamber at Bell Labs using the same instruments and settings, and the anechoic measurement served as the approximated ground truth for evaluating the performance of different BCI approaches.

Figure 3 shows the source estimates in the 10 experiments in terms of their normalized correlation to the anechoic measurement. The performance of the proposed BSCI is compared with the beam-forming, the eigenvalue decomposition (Eq. 2), and the convex LS (Eq. 3) approaches. The results of the 10 experiments unanimously support our previous findings in simulations. First, the convex LS approach yielded significantly better source estimates than the eigenvalue decomposition method. Second, the proposed BSCI approach, which incorporates the convex LS formulation with the sparse RIR model, yielded the most dramatic results, achieving $85\%$ or higher of normalized correlation in source estimates in most experiments while the LS approach only obtained approximately $70\%$ of normalized correlation.

Figure 4 shows one instance of filter and source estimates. The estimated filters have about 2000 zeros out of totally 3072 coefficients, and thus they are sparse. This observation experimentally validates our hypothesis of the sparse RIR models, namely, an acoustic RIR can be modeled by a sparse FIR filter. The source estimate shown in Fig. 4 vividly illustrates the convolution and dereverberation process. It only plots a small segment to reveal greater details. As we see, the anechoic measurement was clean and had clear harmonic structure; the signal recorded in the reverberant room was smeared by echoes during the convolution process; and then, the dereverberation using our BSCI approach deblurred the signal and recovered the underlying harmonic structure.

## 4  Discussion

We propose a *blind sparse channel identification* (BSCI) approach for speech dereverberation. It consists of three important components. The first is the *sparse RIR model*, which effectively resolves solution degeneracies and robustly models real acoustic environments. The second is the *convex formulation*, which guarantees global convergence of the proposed BSCI algorithm. And the third is the *Bayesian $l_1$-norm sparse learning* scheme that infers the optimal regularization parameters for deriving optimally sparse solutions. The results demonstrate that the proposed BSCI approach holds the potential to solve the speech dereverberation problem in real acoustic environments, which has been recognized as a very difficult problem in signal processing. The acoustic data used in this paper are available at http://www.seas.upenn.edu/∼linyuanq/Research.html.

Our future work includes side-by-side comparison between our BSCI approach and existing source statistics based BCI approaches. Our goal is to build a uniform framework that combines various prior knowledge about acoustic systems for best solving the speech dereverberation problem.

## References

[1] T. Nakatani, M. Miyoshi, and K. Kinoshita, "One microphone blind dereverberation based on quasi-periodicity of speech signals," in *NIPS 16*. 2004.

[2] A. Hyvarinen, J. Karhunen, and E. Oja, *Independent Component Analysis*, New York, NY: John Wiley and Sons, 2001.

[3] H. Attias, J. C. Platt, A. Acero, and L. Deng, "Speech denoising and dereverberation using probabilistic models," in *NIPS 13*, 2000.

[4] L. Tong, G. Xu, and T. Kailath, "Blind identification and equalization based on second-order statistics: A time domain approach," *IEEE Trans. Information Theory*, vol. 40, no. 2, pp. 340–349, 1994.

[5] J. B. Allen and D. A. Berkley, "Image method for efficiently simulating small-room acoustics," *J. Acoustical Society America*, vol. 65, pp. 943–950, 1979.

[6] D. L. Duttweiler, "Proportionate normalized least-mean-squares adaptation in echo cancelers," *IEEE Trans. Speech Audio Processing*, vol. 8, pp. 508–518, 2000.

[7] Y. Lin and D. D. Lee, "Bayesian $L_1$-norm sparse learning," in *Proc. ICASSP*, 2006.

[8] S. S. Chen, D. L. Donoho, and M. A. Saunders, "Atomic decomposition by basis pursuit," *SIAM J. Scientific Computing*, vol. 20, no. 1, pp. 33–61, 1998.

[9] S. J. Wright, *Primal-Dual Interior Point Methods*, Philadelphia, PA: SIAM, 1997.

[10] D. M. Malioutov, M. Cetin, and A. S. Willsky, "Homotopy continuation for sparse signal representation," in *Proc. ICASSP*, 2005.

[11] D.A. Harville, "Maximum likelihood approaches to variance component estimation and to related problems," *J. American Statistical Association*, vol. 72, pp. 320–338, 1977.

[12] M. Miyoshi and Y. Kaneda, "Inverse filtering of room acoustics," *IEEE Trans. Acoustics, Speech, and Signal Processing*, vol. 36, no. 2, pp. 145–152, 1988.

